# Deriving Receptive Fields Using An Optimal Encoding Criterion

**Ralph Linsker**
IBM T. J. Watson Research Center
P. O. Box 218, Yorktown Heights, NY 10598

## Abstract

An information-theoretic optimization principle ('infomax') has previously been used for unsupervised learning of statistical regularities in an input ensemble. The principle states that the input-output mapping implemented by a processing stage should be chosen so as to maximize the average mutual information between input and output patterns, subject to constraints and in the presence of processing noise. In the present work I show how infomax, when applied to a class of nonlinear input-output mappings, can under certain conditions generate optimal filters that have additional useful properties: (1) Output activity (for each input pattern) tends to be concentrated among a relatively small number of nodes. (2) The filters are sensitive to higher-order statistical structure (beyond pairwise correlations). If the input features are localized, the filters' receptive fields tend to be localized as well. (3) Multiresolution sets of filters with subsampling at low spatial frequencies – related to pyramid coding and wavelet representations – emerge as favored solutions for certain types of input ensembles.

## 1  INTRODUCTION

In unsupervised network learning, the development of the connection weights is influenced by statistical properties of the ensemble of input vectors, rather than by the degree of mismatch between the network's output and some 'desired' output. An implicit goal of such learning is that the network should transform the input so that salient features present in the input are represented at the output in a

more useful form. This is often done by reducing the input dimensionality in a way that preserves the high-variance components of the input (e.g., principal component analysis, Kohonen feature maps).

The principle of maximum information preservation ('infomax') is an unsupervised learning strategy that states (Linsker 1988): From a set of allowed input-output mappings (e.g., parametrized by the connection weights), choose a mapping that maximizes the (ensemble-averaged) Shannon information that the output vector conveys about the input vector, in the presence of noise. Such a mapping maximizes the ensemble-averaged mutual information (MI) between input and output.

This paper (a) summarize earlier results on infomax solutions for linear networks, (b) identifies some limitations of these solutions (ways in which very different filter sets are equally optimal from the infomax standpoint), and (c) shows how, by adding a small nonlinearity to the network, one can remove these limitations and at the same time improve the utility of the output representations. We show that infomax, acting on the modified network, tends to favor sparsely coded representations and (depending on the input ensemble) sets of filters that span multiple resolution scales (related to wavelets and 'pyramid coding').

## 2    INFOMAX IN LINEAR NETWORKS

For definiteness and brevity, we consider a linear network having a particular type of noise model and input statistical properties. For a more detailed discussion of related models see (Linsker 1989).

Since the computation of the MI (which involves the output entropy) is in general intractable for continuous-valued output vectors, previous work (and the present paper) makes use of a surrogate MI, which we will call the 'as-if-Gaussian' MI. This quantity is, by definition, computed as though the output vectors comprised a multivariate Gaussian distribution having the same mean and covariance as the actual distribution of output vectors. Although expedient, this substitution has lacked a principled justification. The Appendix shows that, under certain conditions, using this 'surrogate MI' (and not the full MI) is indeed appropriate and justified.

Denote the input vector by $S \equiv \{S_i\}$ ($S_i$ is the activity at input node $i$), the output vector by $Z \equiv \{Z_n\}$, the matrix of connection weights by $C \equiv \{C_{ni}\}$, noise at the input nodes by $N \equiv \{N_i\}$, and noise at the output nodes by $\nu \equiv \{\nu_n\}$. Then our processing model is, in matrix form, $Z = C(S + N) + \nu$. Assume that $N$ and $\nu$ are Gaussian random variables, $\langle S \rangle = \langle N \rangle = \langle \nu \rangle = 0$, $\langle SN^T \rangle = \langle S\nu^T \rangle = \langle N\nu^T \rangle = 0$, and, for the covariance matrices, $\langle SS^T \rangle = Q$, $\langle NN^T \rangle = \eta I$, $\langle \nu\nu^T \rangle = \beta I'$. (Angle brackets denote an ensemble average, superscript $T$ denotes transpose, and $I$ and $I'$ denote unit matrices on the input and output spaces, respectively.) In general, $\mathrm{MI} = H_Z - \langle H_{Z|S} \rangle$ where $H_Z$ is the output entropy and $H_{Z|S}$ is the entropy of the output for given $S$. Replacing MI by the 'as-if-Gaussian' MI means replacing $H_Z$ by the expression for the entropy of a multivariate Gaussian distribution, which is (apart from an irrelevant constant term) $H_Z^G = (1/2) \ln \det Q'$, where $Q' \equiv \langle ZZ^T \rangle = CQC^T + \eta CC^T + \beta I'$ is the output covariance. Note that, when $S$ is fixed, $Z = CS + (CN + \nu)$ is a Gaussian distribution centered on $CS$, so that we have $\langle H_{Z|S} \rangle = (1/2) \ln \det Q''$ where $Q'' = \langle (CN + \nu)(CN + \nu)^T \rangle = \eta CC^T + \beta I'$. Therefore the

'as-if-Gaussian' MI is

$$\text{MI}' = (1/2)[\ln \det Q' - \ln \det Q''].  \qquad (1)$$

The variance of the output at node $n$ (prior to adding noise $\nu_n$) is $V_n = \langle [C(S + N)]_n^2 \rangle = (CQC^T + \eta CC^T)_{nn}$. We will constrain the dynamic range of each output node (limiting the number of output values that can be discriminated from one another in the presence of output noise) by requiring that $V_n = 1$ for each $n$. Subject to this constraint, we are to find a matrix $C$ that maximizes MI'. For a local Hebbian algorithm that accomplishes this maximization, see (Linsker 1992). Here, in order to proceed analytically, we consider a special case of interest.

Suppose that the input statistics are shift-invariant, so that the covariance $\langle S_i S_j \rangle$ is a function of $(j - i)$. We then use a shift-invariant filter Ansatz, $C_{ni} \equiv C(i - n)$. Infomax then determines the optimal filter gain as a function of spatial frequency; i.e., the magnitude of the Fourier components $c(k)$ of $C(i - n)$. The derivation is summarized below.

Denote by $q(k)$, $q'(k)$, and $q''(k)$ the Fourier transforms of $Q(j - i)$, $Q'(m - n)$, and $Q''(m - n)$ respectively. Since $Q' = CQC^T + \eta CC^T + \beta I'$, therefore $q'(k) = [q(k) + \eta] \mid c(k) \mid^2 + \beta$. Similarly, $q''(k) = \eta \mid c(k) \mid^2 + \beta$. We obtain $\text{MI}' = (1/2)\Sigma_k [\ln q'(k) - \ln q''(k)]$. Each node's output variance $V_n$ is equal to $V = (1/K)\Sigma_k [q(k) + \eta] \mid c(k) \mid^2$ where $K$ is the number of terms in the sum over $k$.

To maximize MI' subject to the constraint on $V$ we use the Lagrange multiplier method; that is, we maximize $\text{MI}'' \equiv \text{MI}' + \mu(V - 1)$ with respect to each $\mid c(k) \mid^2$. This yields an equation for each $k$ that is quadratic in $\mid c(k) \mid^2$. The unique solution is

$$(\eta/\beta) \mid c(k) \mid^2 = -1 + \frac{q(k)}{2[q(k) + \eta]}\left\{ 1 + [1 - \frac{2\eta K}{\mu \beta q(k)}]^{1/2} \right\}  \qquad (2)$$

if the RHS is positive, and zero otherwise. The Lagrange multiplier $\mu(< 0)$ is chosen so that the $\{\mid c(k) \mid\}$ satisfy $V = 1$.

Starting from a differently-stated goal (that of reducing redundancy subject to a limit on information loss), which turns out to be closely related to infomax, (Atick & Redlich 1990a) found an expression for the optimal filter gain that is the same as that of Eq. 2 except for the choice of constraint.

Filter properties found using this approach are related to those found in early stages of biological sensory processing. Smoothing and bandpass (contrast-enhancing) filters emerge as infomax solutions (Linsker 1989, Atick & Redlich 1990a) in certain cases, and good agreement with retinal contrast sensitivity measurements has been found (Atick & Redlich 1990b).

Nonetheless, the value of the infomax solution Eq. 2 is limited in two important ways. First, the phases of the $\{c(k)\}$ are left undetermined. Any choice of phases is equally good at maximizing MI' in a linear network. Thus the real-space response function $C(i - n)$, which determines the receptive field properties of the output nodes, is nonunique (and indeed may be highly nonlocalized in space).

Second, it is useful to extend the solution Ansatz to allow a number of different filter types $a = 1, \ldots, A$ at each output site, while continuing to require that each type

satisfy the shift-invariance condition $C_{ni}(a) \equiv C(i-n;a)$. For example, one may want to model a topographic 'retinocortical' mapping in which each patch of cortex (each 'site') contains multiple filter types, yet each patch carries out the same set of processing functions on its input. For this Ansatz, one again obtains Eq. 2 (derivation omitted here), but with $|c(k)|^2$ on the LHS replaced by $\Sigma_a \rho(a)|c(k;a)|^2$, where $c(k;a)$ is the F.T. of $C(i-n;a)$, and $\rho(a)$ is the fraction of the total number of filters (at each site) that are of type $a$. The partitioning of the overall (sum-squared) gain among the multiple filter types is thus left undetermined.

The higher-order statistical structure of the input (beyond covariance) is not being exploited by infomax in the above analysis, because (1) the network is linear and (2) only pairwise correlations among the output activities enter into MI'. We shall show that if we make the network even mildly nonlinear, MI' is no longer independent of the choice of phases or of the partitioning of gain among multiple filter types.

# 3    NETWORK WITH WEAK NONLINEARITY

We consider the weakly nonlinear input-output relation $Z_n = U_n + \epsilon U_n^3 + \Sigma_i C_{ni} N_i + \nu_n$, where $U_n \equiv \Sigma_i C_{ni} S_i$, for small $\epsilon$. This differs from the linear network analyzed above by the term in $U_n^3$. (For simplicity, terms nonlinear in the noise are not included.) The cubic term increases the signal-to-noise ratio selectively when $U_n$ is large in absolute value. We maximize MI' as defined in Eq. 1.

Heuristically, the new term will cause infomax to favor solutions in which some output nodes have large (absolute) activity values, over solutions in which all output nodes have moderate activities. The output layer can thus encode information about the input vector (e.g., signal the presence of a feature) via the high activity of a small number of nodes, rather than via the particular activity values of many nodes. This has several (interrelated) potential advantages. (1) The concentration of activity among fewer nodes is a type of sparse coding. (2) The resulting output representation may be more resistant to noise. (3) The presence of a feature can be signaled to a later processing stage using fewer connections. (4) Since the particular nodes that have high activity depend upon the input vector, this type of mapping transforms a set of continuous-valued inputs at each site into a partially place-coded representation. A model of this sort may thus be useful for understanding better the formation of place-coded representations in biological systems.

## 3.1    MATHEMATICAL DETAILS

This section may be skipped without loss of continuity. In matrix form, $U \equiv CS$, $W_n \equiv U_n^3$ for each $n$, and $Z = U + \epsilon W + CN + \nu$. Keeping terms through first order in $\epsilon$, the output covariance is $Q' \equiv \langle ZZ^T \rangle = CQC^T + \eta CC^T + \beta I' + \epsilon F$, where $F \equiv \langle WU^T \rangle + \langle UW^T \rangle$. [As an aside, $F_{nm} = \langle U_n U_m (U_n^2 + U_m^2) \rangle$ resembles the covariance $\langle U_n U_m \rangle$, except that presentations having large $U_n^2 + U_m^2$ are given greater weight in the ensemble average.] For shift-invariant input statistics and one filter type $C_{ni} \equiv C(i-n)$, taking the Fourier transform yields $q'(k) = [q(k)+\eta]|c(k;a)|^2 + \beta + \epsilon f(k)$ where $f(k)$ is the F.T. of $F(m-n) \equiv F_{nm}$. So $\ln \det Q' = \Sigma_k \ln q'(k) = \Sigma \ln\{[q(k)+\eta]|c(k)|^2 + \beta\} + \epsilon \Sigma g(k)$ where $g(k) \equiv [f(k)/\{[q(k)+\eta]|c(k;a)|^2 + \beta\}]$. Using a Lagrange multiplier as before, the quantity to be maximized is MI'' =

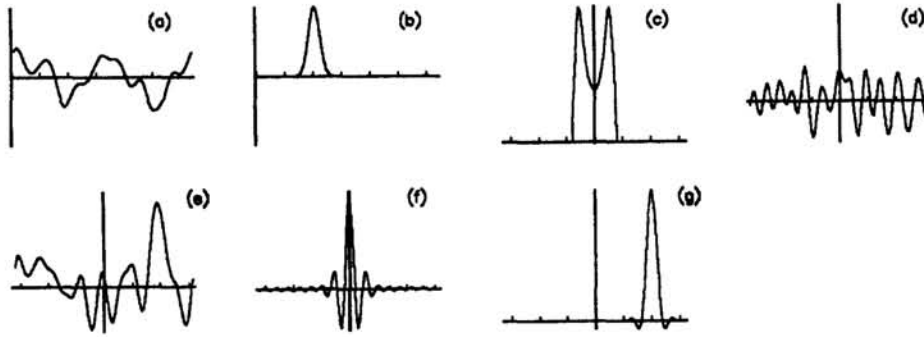

Figure 1: Breaking of phase degeneracy. See text for discussion.

$\text{MI}''(\epsilon = 0) + (\epsilon/2)\Sigma g(k)$.

Now suppose there are multiple filter types $a = 1, \ldots, A$ at each output site. For each $k$ define $d(k)$ to be the $A \times A$ matrix whose elements are: $d(k)_{ab} \equiv [q(k) + \eta]c(k;a)c^*(k;b) + [\beta/\rho(a)]\delta_{ab}$ where $\delta_{ab}$ is the Kronecker delta. Also define $f(k)$ to be the $A \times A$ matrix each of whose elements $f(k)_{ab}$ is the F.T. of $F(m-n;a,b)$ where $F(m-n;a,b) = \langle U_n(a)W_m(b)\rangle + \langle W_n(a)U_m(b)\rangle$. Then the $O(\epsilon)$ part of $\text{MI}''$ is: $(\epsilon/2)\Sigma_k\text{Tr}\{[d(k)]^{-1}f(k)\}$. Note that $[d(k)]^{-1}$ is the inverse of the matrix $d(k)$, and that 'Tr' denotes the trace. [Outline of derivation: In the basis defined by the Fourier harmonics, $Q'$ is block diagonal (one $A \times A$ block for each $k$). So $\ln \det Q' = \Sigma_k \ln \det q'(k)$ where each $q'(k)$ is an $A \times A$ matrix of the form $q'_1(k) + \epsilon q'_2(k)$. Expanding $\ln \det q'(k)$ through $O(\epsilon)$ yields the stated result.]

The infomax calculation to lowest order in $\epsilon$ [i.e., $O(\epsilon^0)$] is the same as for the linear network. Here, for simplicity, we determine the sum-squared gain, $\Sigma_a \rho(a)|c(k;a)|^2$, as in the linear case; then seek to maximize the new term, of $O(\epsilon)$, subject to this constraint on the value of the sum-squared gain. How the nonlinear term breaks phase and gain-apportionment degeneracies is of interest here; a small $O(\epsilon)$ correction to the sum-squared gain is not.

## 4  ILLUSTRATIVE RESULTS

Two examples will show how adding the nonlinear perturbative term to the network's output breaks a degeneracy among different filter solutions. In each case the input space is a one-dimensional 'retina' with wraparound.

### 4.1  BREAKING THE PHASE DEGENERACY

In this example (see Figure 1) there is one filter type at each output site. We consider two types of input ensembles: (1) Each input vector (Fig. 1a shows one example) is drawn from a multivariate Gaussian distribution (so there is no higher-order statistical structure beyond pairwise correlations). The input covariance matrix $Q(j - i)$ is a Gaussian function of the distance between the sites. (2) Each input

vector is a random sum of Gaussian 'bumps': $S_i = \Sigma_j a_j [s(i-j) - s_0]$ where $s(i-j)$ is a Gaussian (shown in Fig. 1b for j=20; there are 64 nodes in all); $s_0$ is the mean value of $s(i-j)$; and each $a_j$ is independently and randomly chosen (with constant probability) to be 1 or 0. This ensemble does have higher-order structure, with each input presentation being characterized by the presence of localized features (the bumps) at particular locations.

The infomax solution for $| c(k) |^2$ is plotted versus spatial frequency $k$ in Fig. 1c for a particular choice of noise parameters $(\eta, \beta)$. As stated earlier, MI' for a linear network is indifferent to the phases of the Fourier components $\{c(k)\}$. A particular random choice of phases produces the real-space filter $C(i-n)$ shown in Fig. 1d, which spans the entire 'retina.' Setting all phases to zero produces the localized filter shown in Fig. 1f. If the Gaussian 'bump' of Fig. 1b is presented as input to a network of filters each of which is a shifted version of Fig. 1d, the linear response of the network (i.e., the convolution of the 'bump' with the filter) is shown in Fig. 1e. Replacing the filter of Fig. 1d by that of Fig. 1f, but keeping the input the same, produces the output response shown in Fig. 1g.

The cubic nonlinearity causes MI' to be larger for the filter of Fig. 1f than for that of Fig. 1d. Heuristically, if we focus on the diagonal elements of the output covariance $Q'$, the nonlinear term is $2\epsilon \langle U_n^4 \rangle$. Maximizing MI' favors increasing this term (subject to a constraint on output variance) hence favors filter solutions for which the $U_n$ distribution is non-Gaussian with a preponderance of large values. Projection pursuit methods also use a measure of the non-Gaussianity of the output distribution to construct filters that extract 'interesting' features from high-dimensional data (cf. Intrator 1992).

## 4.2 BREAKING THE PARTITIONING DEGENERACY FOR MULTIPLE FILTER TYPES

In this example (see Fig. 2), the input ensemble comprises a set of self-similar patterns (each is a sine-Gabor 'ripple' as in Fig. 2a) that are related by translation and dilation (scale change over a factor of 80). Figure 2b shows the input power spectrum vs. $k$; the scaling region goes as $1/k$. Figure 2c shows the infomax solution for the gain $| c(k; a) |$ vs. $k$ when there is just one filter type. When the input SNR is large (as in the scaling region) the infomax filters 'whiten' the output; note the flat portion of the output power spectrum (Fig. 2d). [We modify the infomax solution by extending the power-law form of $| c(k) |$ to low $k$ (dotted line in Figs. 2c,d). This avoids artifacts resulting from the rapid increase in $| c(k) |$, which is in turn caused by our having omitted low-$k$ patterns from the input ensemble for reasons of numerical efficiency.] The dotted envelope curve in Figure 2e shows the sum-squared gain $\Sigma_a \rho(a) | c(k) |^2$ when multiple filter types $a$ are allowed. The quantity plotted is just the square of that shown in Fig. 2c, but on a linear rather than log-log plot (note values greater than 5 are cut off to save space).

The network nonlinearity has the following effect. We first allow two filter types to share the overall gain. Optimizing MI' over various partitionings, we find that infomax favors a crossover between filter types at $k \approx 400$. Allowing three, then four, filter types produces additional crossovers at lower $k$. For an Ansatz in which each filter's share of the sum-squared gain is tapered linearly near its cutoff frequencies,

the best solution found for each $\rho(a) \mid c(k) \mid^2$ is shown in Fig. 2e (semilog plot vs. $k$). Figure 2f plots the corresponding $\mid c(k;a) \mid$ vs. $k$ on a linear scale. Note that the three lower-$k$ filters appear roughly self-similar. (The peak in the highest-$k$ filter is an artifact due to the cutoff of the input ensemble at high $k$.) The four real-space filters $C(i-n;a)$ are plotted vs. $(i-n)$ in Fig. 2g [phases chosen to make $C(i-n;a)$ antisymmetric].

The resulting filters span multiple resolution scales. The density $\rho(a)$ is less for the lower-frequency filters (spatial subsampling). When more filter types are allowed, the increase in MI′ becomes progressively less. Although in our model the filters are present with density $\rho$ at each output site, a similar MI′ is obtained if one spaces adjacent filters of type $a$ by a distance $\propto 1/\rho(a)$. The resulting arrangement of filters resembles the 'tiling' of the joint space and spatial-frequency domain that is used in wavelet and 'pyramid coding' approaches to image processing. [The infomax filters overlap, rather than disjointly tiling the $(x, k)$ domain.]

Using the infomax method, the region of $(x, k)$ space spanned by an optimal filter has an aspect ratio that depends upon the relative distances – along the $x$ and $k$ axes – over which the input feature is 'coherent' (possesses higher-order correlations). One may thus be able to use infomax to predict relationships between statistical measures of coherence in natural scenes and observed $(x, k)$ aspect ratios for, e.g., orientation-selective cells. See (Field 1989) for a discussion of this issue that is not based on infomax.

## 5  APPENDIX: HEURISTIC JUSTIFICATION FOR USING A SURROGATE, 'AS-IF-GAUSSIAN,' MUTUAL INFORMATION

The mutual information between input $S$ and output $Z$ is MI $= \int dS dZ P_{SZ} \ln(P_{SZ}/P_S P_Z) = \int dS P_S \mathrm{KD}(P_{Z|S}; P_Z)$ where $\mathrm{KD}(P_{Z|S}; P_Z) = \int dZ P_{Z|S} \ln(P_{Z|S}/P_Z)$ is a Kullback divergence. So, maximizing MI means maximizing the average (over $S$) of $\mathrm{KD}(P_{Z|S}; P_Z)$.

What does the KD represent? Suppose that the network has somehow learned the distribution $P_Z$. Before being presented with a particular input $S$, the network 'expects' an output vector drawn from $P_Z$. The actual output response to $S$, however, is a vector drawn from $P_{Z|S}$. The KD measures the 'surprise' (i.e., the amount of information gained) upon seeing the actual distribution $P_{Z|S}$ when one expected $P_Z$. Infomax maximizes this average 'surprise.'

However, the network cannot in general have access to the full distribution $P_Z$, which contains far too much information (including all higher-order statistics) to be stored in the connections and nodes of the network. Let us suppose for definiteness that the system remembers only the mean and the covariance matrix of $Z$. Define $P_Z^G$ to be the multivariate Gaussian distribution that has the same mean and covariance as $P_Z$. Then we may think of the system as a priori 'expecting' the output vector to be drawn from the distribution $P_Z^G$.

We accordingly modify the principle so that we maximize the average (over $S$) of $\mathrm{KD}(P_{Z|S}; P_Z^G)$ (note the superscript $G$). This equals

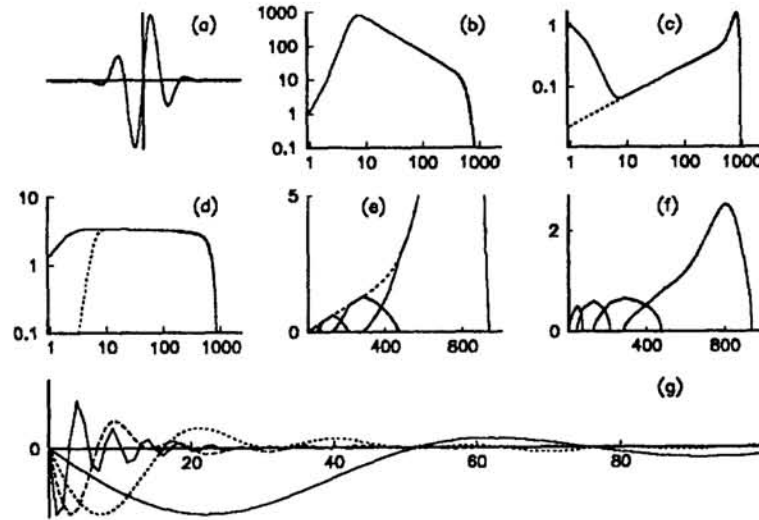

Figure 2: Partitioning among multiple filter types. See text.

$\int dS P_S \int dZ P_{Z|S} \ln(P_{Z|S}/P_Z^G) = \langle -H_{Z|S} \rangle_S - \int dZ P_Z \ln P_Z^G$ (where $H$ denotes entropy). Using a property of the Gaussian distribution, we have $-\int dZ P_Z \ln P_Z^G = -\int dZ P_Z^G \ln P_Z^G = H_Z^G$. We conclude that the average of KD equals $H_Z^G - \langle H_{Z|S} \rangle_S$, which is exactly equal to the surrogate 'as-if-Gaussian' MI defined preceding Eq. 1. This argument provides a principled justification for using the surrogate MI, when the system has stored information about the output vectors' mean and covariance, but not about higher-order statistics.

## References

J. J. Atick & A. N. Redlich. (1990a) Towards a theory of early visual processing. *Neural Computation* 2:308-320.

J. J. Atick & A. N. Redlich. (1990b) Quantitative tests of a theory of retinal processing: contrast sensitivity curves. Inst. Adv. Study IASSNS-HEP-90/51.

D. J. Field. (1989) What the statistics of natural images tell us about visual coding. In *Proc. SPIE* **1077**:269-276.

N. Intrator. (1992) Feature extraction using an unsupervised neural network. *Neural Computation* 4:98-107.

R. Linsker. (1988) Self-organization in a perceptual network. *Computer* **21**(3):105-117.

R. Linsker. (1989) An application of the principle of maximum information preservation to linear systems. In D. S. Touretzky (ed.), *Advances in Neural Information Processing Systems 1*, 186-194. San Mateo, CA: Morgan Kaufmann.

R. Linsker. (1992) Local synaptic learning rules suffice to maximize mutual information in a linear network. *Neural Computation* 4(5):691-702.
